# Periodic Finite State Controllers for Efficient POMDP and DEC-POMDP Planning

**Joni Pajarinen**
Aalto University, Department of
Information and Computer Science,
P.O. Box 15400, FI-00076 Aalto, Finland
Joni.Pajarinen@aalto.fi

**Jaakko Peltonen**
Aalto University, Department of Information
and Computer Science, Helsinki Institute for
Information Technology HIIT,
P.O. Box 15400, FI-00076 Aalto, Finland
Jaakko.Peltonen@aalto.fi

## Abstract

Applications such as robot control and wireless communication require planning under uncertainty. Partially observable Markov decision processes (POMDPs) plan policies for single agents under uncertainty and their decentralized versions (DEC-POMDPs) find a policy for multiple agents. The policy in infinite-horizon POMDP and DEC-POMDP problems has been represented as finite state controllers (FSCs). We introduce a novel class of *periodic FSCs*, composed of layers connected only to the previous and next layer. Our periodic FSC method finds a deterministic finite-horizon policy and converts it to an initial periodic infinite-horizon policy. This policy is optimized by a new infinite-horizon algorithm to yield deterministic periodic policies, and by a new expectation maximization algorithm to yield stochastic periodic policies. Our method yields better results than earlier planning methods and can compute larger solutions than with regular FSCs.

## 1 Introduction

Many machine learning applications involve *planning under uncertainty*. Such planning is necessary in medical diagnosis, control of robots and other agents, and in dynamic spectrum access for wireless communication systems. The planning task can often be represented as a reinforcement learning problem, where an action policy controls the behavior of an agent, and the quality of the policy is optimized to maximize a reward function. Single agent policies can be optimized with partially observable Markov decision processes (POMDPs) [1], when the world state is uncertain. Decentralized POMDPs (DEC-POMDPs) [2] optimize policies for multiple agents that act without direct communication, with separate observations and beliefs of the world state, to maximize a joint reward function. POMDP and DEC-POMDP methods use various representations for the policies, such as value functions [3], graphs [4, 5], or finite state controllers (FSCs) [6, 7, 8, 9, 10].

We present a novel efficient method for POMDP and DEC-POMDP planning. We focus on infinite-horizon problems, where policies must operate forever. We introduce a new policy representation: *periodic finite state controllers*, which can be seen as an intelligent restriction which speeds up optimization and can yield better solutions. A periodic FSC is composed of several layers (subsets of states), and transitions are only allowed to states in the next layer, and from the final layer to the first. Policies proceed through layers in a periodic fashion, and policy optimization determines the probabilities of state transitions and action choices to maximize reward. Our work has four main contributions. **Firstly**, we introduce an improved optimization method for standard finite-horizon problems with FSC policies by *compression*. **Secondly**, we give a method to transform the finite-horizon FSC into an initial infinite-horizon periodic FSC. **Thirdly**, we introduce compression to the periodic FSC. **Fourthly**, we introduce an expectation-maximization (EM) training algorithm for planning with periodic FSCs. We show that the resulting method performs better than earlier DEC-

POMDP methods and POMDP methods with a restricted-size policy and that use of the periodic FSCs enables computing larger solutions than with regular FSCs. Online execution has complexity $\mathcal{O}(const)$ for deterministic FSCs and $\mathcal{O}(\log(FSC\ layer\ width))$ for stochastic FSCs.

We discuss existing POMDP and DEC-POMDP solution methods in Section 2 and formally define the infinite-horizon (DEC-)POMDP. In Section 3 we introduce the novel concept of periodic FSCs. We then describe the stages of our method: improving finite-horizon solutions, transforming them to periodic infinite-horizon solutions, and improving the periodic solutions by a novel EM algorithm for (DEC-)POMDPs (Section 3.2). In Section 4 we show the improved performance of the new method on several planning problems, and we conclude in Section 5.

## 2 Background

Partially observable Markov decision processes (POMDPs) and decentralized POMDPs (DEC-POMDPs) are model families for decision making under uncertainty. POMDPs optimize policies for a single agent with uncertainty of the environment state while DEC-POMDPs optimize policies for several agents with uncertainty of the environment state and each other's states. Given the actions of the agents the environment evolves according to a Markov model. The agents' policies are optimized to maximize the expected reward earned for actions into the future. In infinite-horizon planning the expected reward is typically discounted to emphasize current and near-future actions. Computationally POMDPs and DEC-POMDPs are complex: even for finite-horizon problems, finding solutions is in the worst case PSPACE-complete for POMDPs and NEXP-complete for DEC-POMDPs [11].

For infinite-horizon DEC-POMDP problems, state of the art methods [8, 12] store the policy as a stochastic finite state controller (FSC) for each agent which keeps the policy size bounded. The FSC parameters can be optimized by expectation maximization (EM) [12]. An advantage of EM is that it can be adapted to for example continuous probability distributions [7] or to take advantage of factored problems [10]. Alternatives to EM include formulating FSC optimization as a non-linear constraint satisfaction (NLP) problem solvable by an NLP solver [8], or iteratively improving each FSC by linear programming with other FSCs fixed [13]. Deterministic FSCs with a fixed size could also be found by a best-first search [14]. If a DEC-POMDP problem has a specific goal state, then a goal-directed [15] approach can achieve good results. The NLP and EM methods have yielded the best results for the infinite-horizon DEC-POMDP problems. In a recent variant called *mealy NLP* [16], the NLP based approach to DEC-POMDPs is adapted to FSC policies represented by Mealy machines instead of traditional Moore machine representations. In POMDPs, Mealy machine based controllers can achieve equal or better solutions than Moore controllers of the same size.

This paper recognizes the need to improve general POMDP and DEC-POMDP solutions. We introduce an approach where FSCs have a periodic layer structure, which turns out to yield good results.

### 2.1 Infinite-horizon DEC-POMDP: definition

The tuple $\langle \{\alpha_i\}, S, \{A_i\}, P, \{\Omega_i\}, O, R, b_0, \gamma \rangle$ defines an infinite-horizon DEC-POMDP problem for $N$ agents $\alpha_i$, where $S$ is the set of environment states and $A_i$ and $\Omega_i$ are the sets of possible actions and observations for agent $\alpha_i$. A POMDP is the special case when there is only one agent. $P(s'|s, \vec{a})$ is the probability to move from state $s$ to $s'$, given the actions of all agents (jointly denoted $\vec{a} = \langle a_1, \dots, a_N \rangle$). The observation function $O(\vec{o}|s', \vec{a})$ is the probability that the agents observe $\vec{o} = \langle o_1, \dots, o_N \rangle$, where $o_i$ is the observation of agent $i$, when actions $\vec{a}$ were taken and the environment transitioned to state $s'$. The initial state distribution is $b_0(s)$. $R(s, \vec{a})$ is the real-valued reward for executing actions $\vec{a}$ in state $s$. For brevity, we denote transition probabilities given the actions by $P_{s's\vec{a}}$, observation probabilities by $P_{\vec{o}s'\vec{a}}$, reward functions by $R_{s\vec{a}}$, and the set of all agents other than $i$ by $\bar{i}$. At each time step, agents perform actions, the environment state changes, and agents receive observations. The goal is to find a joint policy $\pi$ for the agents that maximizes expected discounted infinite-horizon reward $E\left[\sum_{t=0}^{\infty} \gamma^t R_{s(t)\vec{a}(t)} | \pi\right]$, where $\gamma$ is the discount factor, and $s(t)$ and $\vec{a}(t)$ are the state and action at time $t$, and $E[\cdot|\pi]$ denotes expected value under policy $\pi$. Here, the policy is stored as a set of stochastic finite state controllers (FSCs), one for each agent. The FSC of agent $i$ is defined by the tuple $\langle Q_i, \nu_{q_i}, \pi_{a_i q_i}, \lambda_{q'_i q_i o_i} \rangle$, where $Q_i$ is the set of FSC nodes $q_i$, $\nu_{q_i}$ is the initial distribution $P(q_i)$ over nodes, $\pi_{a_i q_i}$ is the probability $P(a_i|q_i)$ to perform action $a_i$ in node $q_i$, and $\lambda_{q'_i q_i o_i}$ is the probability $P(q'_i|q_i, o_i)$ to transition from node $q_i$ to node $q'_i$ when

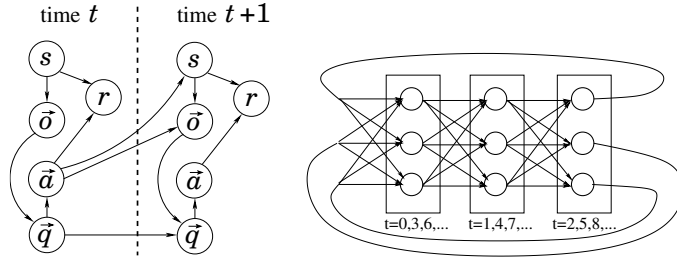

Figure 1: **Left:** influence diagram for a DEC-POMDP with finite state controllers $\vec{q}$, states $s$, joint observations $\vec{o}$, joint actions $\vec{a}$ and reward $r$ (given by a reward function $R(s, \vec{a})$). A dotted line separates two time steps. **Right:** an example of the new *periodic finite state controller*, with three layers and three nodes in each layer, and possible transitions shown as arrows. The controller controls one of the agents. Which layer is active depends only on the current time; which node is active, and which action is chosen, depend on transition probabilities and action probabilities of the controller.

observing $o_i$. The current FSC nodes of all agents are denoted $\vec{q} = \langle q_1, \ldots, q_N \rangle$. The policies are optimized by optimizing the parameters $\nu_{q_i}$, $\pi_{a_i q_i}$, and $\lambda_{q_i' q_i o_i}$. Figure 1 (left) illustrates the setup.

## 3 Periodic finite state controllers

State-of-the-art algorithms [6, 13, 8, 12, 16] for optimizing POMDP/DEC-POMDP policies with restricted-size FSCs find a local optimum. A well-chosen FSC initialization could yield better solutions, but initializing (compact) FSCs is not straightforward: one reason is that dynamic programming is difficult to apply on generic FSCs. In [17] FSCs for POMDPs are built using dynamic programming to add new nodes, but this yields large FSCs and cannot be applied on DEC-POMDPs as it needs a piecewise linear convex value function. Also, general FSCs are irreducible, so a probability distribution over FSC nodes is not sparse over time even if a FSC starts from a single node. This makes computations with large FSCs difficult and FSC based methods are limited by FSC size. We introduce *periodic FSCs*, which allow the use of much larger controllers with a small complexity increase, efficient FSC initialization, and new dynamic programming algorithms for FSCs.

A periodic FSC is composed of $M$ *layers of controller nodes*. Nodes in each layer are connected only to nodes in the next layer: the first layer is connected to the second, the second layer to the third and so on, and the last layer is connected to the first. The *width* of a periodic FSC is the number of controller nodes in a layer. Without loss of generality we assume all layers have the same number of nodes. A single-layer periodic FSC equals an ordinary FSC. A periodic FSC has different action and transition probabilities for each layer. $\pi_{a_i q_i}^{(m)}$ is the layer $m$ probability to take action $a_i$ when in node $q_i$, and $\lambda_{q_i' q_i o_i}^{(m)}$ is the layer $m$ probability to move from node $q_i$ to $q_i'$ when observing $o_i$. Each layer connects only to the next one, so the policy cycles periodically through each layer: for $t \geq M$ we have $\pi_{a_i q_i}^{(t)} = \pi_{a_i q_i}^{(t \bmod M)}$ and $\lambda_{q_i' q_i o_i}^{(t)} = \lambda_{q_i' q_i o_i}^{(t \bmod M)}$ where 'mod' denotes remainder. Figure 1 (right) shows an example periodic FSC.

We now introduce our method for solving (DEC-)POMDPs with periodic FSC policies. We show that the periodic FSC structure allows efficient computation of deterministic controllers, show how to optimize periodic stochastic FSCs, and show how a periodic deterministic controller can be used as initialization to a stochastic controller. The algorithms are discussed in the context of DEC-POMDPs, but can be directly applied to POMDPs.

### 3.1 Deterministic periodic finite state controllers

In a *deterministic* FSC, actions and node transitions are deterministic functions of the current node and observation. To optimize deterministic periodic FSCs we first compute a non-periodic finite-horizon policy. The finite-horizon policy is transformed into a periodic infinite-horizon policy by connecting the last layer to the first layer and the resulting deterministic policy can then be im-

proved with a new algorithm (see Section 3.1.2). A periodic deterministic policy can also be used as initialization for a stochastic FSC optimizer based on expectation maximization (see Section 3.2).

### 3.1.1 Deterministic finite-horizon controllers

We briefly discuss existing methods for deterministic finite-horizon controllers and introduce an improved finite-horizon method, which we use as the initial solution for infinite-horizon controllers.

State-of-the-art *point based* finite-horizon DEC-POMDP methods [4, 5] optimize a policy graph, with restricted width, for each agent. They compute a policy for a single belief, instead of all possible beliefs. Beliefs over world states are sampled centrally using various action heuristics. Policy graphs are built by dynamic programming from horizon $T$ to the first time step. At each time step a policy is computed for each policy graph node, by assuming that the nodes all agents are in are associated with the same belief. In a POMDP, computing the deterministic policy for a policy graph node means finding the best action, and the best connection (best next node) for each observation; this can be done with a direct search. In a DEC-POMDP this approach would go through all combinations of actions, observations and next nodes of all agents: the number of combinations grows exponentially with the number of agents, so direct search works only for simple problems. A more efficient way is to go through all action combinations, for each action combination sample random policies for all agents, and then improve the policy of each agent in turn while holding the other agents' policies fixed. This is not guaranteed to find the best policy for a belief, but has yielded good results in the Point-Based Policy Generation (PBPG) algorithm [5].

We introduce a new algorithm which improves on [5]. PBPG used linear programming to find policies for each agent and action-combination, but with a fixed joint action and fixed policies of other agents we can use fast and simple direct search as follows. Initialize the value function $V(s, \vec{q})$ to zero. Construct an initial policy graph for each agent, starting from horizon $t = T$: **(1)** Project the initial belief along a random trajectory to horizon $t$ to yield a sampled belief $b(s)$ over world states. **(2)** Add, to the graph of each agent, a node to layer $t$. Find the best connections to the next layer as follows. Sample random connections for each agent, then for each agent in turn optimize its connection with connections of other agents fixed: for each action-combination $\vec{a}$ and observation connect to the next-layer node that maximizes value, computed using $b(s)$ and the next layer value function; repeat this until convergence, using random restarts to escape local minima. The best connections and action combination $\vec{a}$ become the policy for the current policy graph node. **(3)** Run (1)-(2) until the graph layer has enough nodes. **(4)** Decrease $t$ and run (1)-(3), until $t = 0$.

We use the above-described algorithm for initialization, and then use a new policy improvement approach shown in Algorithm 1 that improves the policy value monotonically: **(1)** Here we do not use a random trajectory for belief projection, instead we project the belief $b_t(s, \vec{q})$ over world states $s$ and controller nodes $\vec{q}$ (agents are initially assumed to start from the first controller node) from time step $t = 0$ to horizon $T$, through the current policy graph; this yields distributions for the FSC nodes that match the current policy. **(2)** We start from the last layer and proceed towards the first. At each layer, we optimize each agent separately: for each graph node $q_i$ of agent $i$, for each action $a_i$ of the agent, and for each observation $o_i$ we optimize the (deterministic) connection to the next layer. **(3)** If the optimized policy at the node (action and connections) is identical to policy $\pi$ of another node in the layer, we sample a new belief over world states, and re-optimize the node for the new belief; if no new policy is found even after trying several sampled beliefs, we try several uniformly random beliefs for finding policies. We also redirect any connections from the previous policy graph layer to the current node to go instead to the node having policy $\pi$; this "compresses" the policy graph without changing its value (in POMDPs the redirection step is not necessary, it will happen naturally when the previous layer is reoptimized). The computational complexity of Algorithm 1 is $\mathcal{O}(2M|Q|^{2N}|A|^N|O|^N|S|^2 + MN|Q|^N|O|^N|A|^N|S|^2 + CN|Q|^2|A|^N|O||S|)$.

Our finite-horizon method *gets rid of the simplifying assumption* that all FSCs are in the same node, for a certain belief, made in [4, 5]. We only assume that for initialization steps, but not in actual optimization. Our optimization monotonically improves the value of a fixed size policy graph and converges to a local optimum. Here we applied the procedure to finite-horizon DEC-POMDPs; it is adapted for improving deterministic infinite-horizon FSCs in Section 3.1.2. We also have two simple improvements: (1) a speedup: [5] used linear programming to find policies for each agent and action-combination in turn, but simple direct search is faster, and we use that; (2) improved duplicate handling: [5] tried sampled beliefs to avoid duplicate nodes, we also try uniformly random

**1** Initialize $V_{T+1}(s, \vec{q}) = 0$
**2** Using current policy project $b_t(s, \vec{q})$ for $1 \le t \le T$
**3** **for** *Time step $t = T$ to $0$* **do**
**4**    **foreach** *Agent $i$* **do**
**5**       **foreach** *Node $q$ of agent $i$* **do**
**6**          **foreach** $a_i$ **do**
**7**             $h_{\vec{o},\vec{q}'}^{a_i} = \sum_{s,s',\vec{q},\vec{a}} P(\vec{o}, s'|s, \vec{a}) b_t(s, \vec{q}) \prod_{j \neq i} P_t(a_j|q_j) P_t(q_j'|q_j, o_j) V_{t+1}(s', \vec{q}')$
**8**             $\forall o_i \; P_t^{a_i}(q_i'|q_i = q, o_i) = \text{argmax}_{P_t^{a_i}(q_i'|q_i=q,o_i)} \sum_{\vec{q}', \{o_j\}_{j \neq i}} P_t^{a_i}(q_i'|q_i = q, o_i) h_{\vec{o},\vec{q}'}^{a_i}$

**9**           $a_i^* = \underset{a_i}{\text{argmax}} \sum_{s,s',\vec{q},\vec{a},\vec{o},\vec{q}'} [b_t(s, \vec{q}) R(s, \vec{a}) \prod_{j \neq i} P_t(a_j|q_j) + \gamma P_t^{a_i}(q_i'|q_i = q, o_i) h_{\vec{o},\vec{q}'}^{a_i}]$

**10**          $P_t(a_i \neq a_i^*|q_i) = 0, \; P_t(a_i = a_i^*|q_i) = 1, \; P_t(q_i'|q_i, o_i) = P_t^{a_i^*}(q_i'|q_i, o_i)$
**11**          **if** *Any node $p$ already has same policy as $q$* **then**
**12**             For each $q_i$ for which $P_{t-1}(q_i' = q|q_i, o_j) = 1$ redirect link to $q_i' = p$
**13**             Sample belief $b(s, q_j = q \forall j)$ and use it to compute new policy by steps 7-13

**14**    $V_t(s, \vec{q}) = R(s, \vec{a}) \prod_i P_t(a_i|q_i) + \gamma \prod_i P_t(q_i'|q_i, o_i) P(s', \vec{o}|s, \vec{a}) V_{t+1}(s', \vec{q}')$

**Algorithm 1**: Monotonic policy graph improvement algorithm

beliefs, and for DEC-POMDPs we redirect previous-layer connections to duplicate nodes. Unlike the recursion idea in [4] our projection approach is guaranteed to improve value at each graph node and find a local optimum.

### 3.1.2 Deterministic infinite-horizon controllers

To initialize an infinite-horizon problem, we transform a deterministic finite-horizon policy graph (computed as in Section 3.1.1) into an infinite-horizon periodic controller by connecting the last layer to the first. Assuming controllers start from policy graph node 1, we compute policies for the other nodes in the first layer with beliefs sampled for time step $M + 1$, where $M$ is the length of the controller period. It remains to compute the (deterministic) connections from the last layer to the first: approximately optimal connections are found using the beliefs at the last layer and the value function projected from the last layer through the graph to the first layer. This approach can yield efficient controllers on its own, but may not be suitable for problems with a long effective horizon.

To optimize controllers further, we give two changes to Algorithm 1 that enable optimization of infinite-horizon policies: **(1)** To compute beliefs $\hat{b}_u(s, \vec{q})$ over time steps $u$ by projecting the initial belief, first determine an effective projection horizon $T_{proj}$. Compute a $Q_{MDP}$ policy [18] (an upper bound to the optimal DEC-POMDP policy) by dynamic programming. As the projection horizon, use the number of dynamic programming steps needed to gather enough value in the corresponding MDP. Compute the belief $b_t(s, \vec{q})$ for each FSC layer $t$ (needed on line 2 of Algorithm 1) as a discounted sum of projected beliefs: $b_t(s, \vec{q}) = \frac{1}{C} \sum_{u \in \{t, t+M, t+2M, \dots; u \le T_{proj}\}} \gamma^u \hat{b}_u(s, \vec{q})$. **(2)** Compute value function $V_t(s, \vec{q})$ for a policy graph layer by backing up (using line 14 of Algorithm 1) $M - 1$ steps from the previous periodic FSC layer to current FSC layer, one layer at a time.

The complexity of one iteration of the infinite-horizon approach is $\mathcal{O}(2M|Q|^{2N}|A|^N|O|^N|S|^2 + M(M-1)N|Q|^N|O|^N|A|^N|S|^2 + MCN|Q|^2|A|^N|O||S|)$. There is no convergence guarantee due to the approximations, but approximation error decreases exponentially with the period $M$.

### 3.2 Expectation maximization for stochastic infinite-horizon controllers

A stochastic FSC provides a solution of equal or larger value [6] compared to a deterministic FSC with the same number of controller nodes. Many algorithms that optimize stochastic FSCs could be adapted to use periodic FSCs; in this paper we adapt the expectation-maximization (EM) approach [7, 12] to periodic FSCs. The adapted version retains the theoretical properties of regular EM, such as monotonic convergence to a local optimum.

In the EM approach [7, 12] the optimization of policies is written as an inference problem: rewards are scaled into probabilities and the policy, represented as a stochastic FSC, is optimized by EM

iteration to maximize the probability of getting rewards. We now introduce an EM algorithm for (DEC-)POMDPs with periodic stochastic FSCs. We build on the EM method for DEC-POMDPs with standard (non-periodic) FSCs by Kumar and Zilberstein [12]; see [7, 12] for more details of non-periodic EM. First, the reward function is scaled into a probability $\hat{R}(r = 1|s, \vec{a}) = (R(s, \vec{a}) - R_{min})/(R_{max} - R_{min})$, where $R_{min}$ and $R_{max}$ are the minimum and maximum rewards possible and $\hat{R}(r = 1|s, \vec{a})$ is the conditional probability for the binary reward $r$ to be 1. The FSC parameters $\theta$ are optimized by maximizing the reward likelihood $\sum_{T=0}^{\infty} P(T)P(r = 1|T, \theta)$ with respect to $\theta$, where the horizon is infinite and $P(T) = (1 - \gamma)\gamma^T$. This is equivalent to maximizing expected discounted reward in the DEC-POMDP. The EM approach improves the policy, i.e. the stochastic periodic finite state controllers, in each iteration. We next describe the E-step and M-step formulas.

In the E-step, *alpha messages* $\hat{\alpha}^{(m)}(\vec{q}, s)$ and *beta messages* $\hat{\beta}^{(m)}(\vec{q}, s)$ are computed for each layer of the periodic FSC. Intuitively, $\hat{\alpha}(\vec{q}, s)$ corresponds to the discount weighted average probability that the world is in state $s$ and FSCs are in nodes $\vec{q}$, when following the policy defined by the current FSCs, and $\hat{\beta}(\vec{q}, s)$ is intuitively the expected discounted total scaled reward, when starting from state $s$ and FSC nodes $\vec{q}$. The alpha messages are computed by projecting an initial nodes-and-state distribution forward, while beta messages are computed by projecting reward probabilities backward. We compute separate $\hat{\alpha}^{(m)}(\vec{q}, s)$ and $\hat{\beta}^{(m)}(\vec{q}, s)$ for each layer $m$. We use a projection horizon $T = MT_M - 1$, where $MT_M$ is divisible by the number of layers $M$. This means that when we have accumulated enough probability mass in the E-step we still project a few steps in order to reach a valid $T$. For a periodic FSC the forward projection of the joint distribution over world and FSC states from time step $t$ to time step $t + 1$ is $P_t(\vec{q}', s'|\vec{q}, s) = \sum_{\vec{o}, \vec{a}} P_{s's\vec{a}} P_{\vec{o}s'\vec{a}} \prod_i [\pi_{a_i q_i}^{(t)} \lambda_{q_i' q_i o_i}^{(t)}]$. Each $\hat{\alpha}^{(m)}(\vec{q}, s)$ can be computed by projecting a single trajectory forward starting from the initial belief and then adding only messages belonging to layer $m$ to each $\hat{\alpha}^{(m)}(\vec{q}, s)$. In contrast, each $\hat{\beta}^{(m)}(\vec{q}, s)$ has to be projected separately backward, because we don't have a "starting point" similar to the alpha messages. Denoting such projections by $\beta_0^{(m)}(\vec{q}, s) = \sum_{\vec{a}} \hat{R}_{s\vec{a}} \prod_i \pi_{a_i q_i}^{(m)}$ and $\beta_t^{(m)}(\vec{q}, s) = \sum_{s', \vec{q}'} \beta_{t-1}^{(m)}(\vec{q}', s') P_t(\vec{q}', s'|\vec{q}, s)$ the equations for the messages become

$$\hat{\alpha}^{(m)}(\vec{q}, s) = \sum_{t_m=0}^{T_M-1} \gamma^{(m+t_m M)}(1-\gamma)\alpha_{(m+t_m M)}(\vec{q}, s) \text{ and } \hat{\beta}^{(m)}(\vec{q}, s) = \sum_{t=0}^{T} \gamma^t (1-\gamma)\beta_t^{(m)}(\vec{q}, s) . \tag{1}$$

This means that the complexity of the E-step for periodic FSCs is $M$ times the complexity of the E-step for usual FSCs with a total number of nodes equal to the width of the periodic FSC. The complexity increases linearly with the number of layers.

In the M-step we can update the parameters of each layer separately using the alpha and beta messages for that layer, as follows. EM maximizes the expected complete log-likelihood $Q(\theta, \theta^*) = \sum_T \sum_L P(r = 1, L, T|\theta) \log P(r = 1, L, T|\theta^*)$, where $L$ denotes all latent variables: actions, observations, world states, and FSC states, $\theta$ denotes previous parameters, and $\theta^*$ denotes new parameters. For periodic FSCs $P(r = 1, L, T|\theta)$ is

$$P(r = 1, L, T|\theta) = P(T)[\hat{R}_{s\vec{a}}]_{t=T} \left[ \prod_{t=1}^{T} \tau_{\vec{a}\vec{q}}^{(t)} P_{s's\vec{a}} P_{\vec{o}s'\vec{a}} \Lambda_{\vec{q}'\vec{q}\vec{o}t} \right] \left[ \tau_{\vec{a}\vec{q}}^{(0)} b_0(s) \right]_{t=0} \tag{2}$$

where we denoted $\tau_{\vec{a}\vec{q}}^{(t)} = \prod_i \pi_{a_i q_i}^{(t)}$ for $t = 1, \dots, T$, $\tau_{\vec{a}\vec{q}}^{(0)} = \prod_i \pi_{a_i q_i}^{(0)} \nu_{q_i}$, and $\Lambda_{\vec{q}'\vec{q}\vec{o}t} = \prod_i \lambda_{q_i' q_i o_i}^{(t-1)}$.

The log in the expected complete log-likelihood $Q(\theta, \theta^*)$ transforms the product of probabilities into a sum; we can divide the sums into smaller sums, where each sum contains only parameters from the same periodic FSC layer. Denoting $f_{s'sq'\vec{o}\vec{a}m} = P_{s's\vec{a}} P_{\vec{o}s'\vec{a}} \hat{\beta}^{(m+1)}(\vec{q}', s')$, the M-step periodic FSC parameter update rules can then be written as:

$$\nu_{q_i}^* = \frac{\nu_{q_i}}{C_i} \sum_{s, q_{\bar{i}}} \hat{\beta}^{(0)}(\vec{q}, s)\nu_{q_{\bar{i}}} b_0(s) \tag{3}$$

$$\pi_{a_i q_i}^{*(m)} = \frac{\pi_{a_i q_i}^{(m)}}{C_{q_i}} \sum_{s, s', q_{\bar{i}}, \vec{q}', \vec{o}, a_{\bar{i}}} \left\{ \hat{\alpha}^{(m)}(\vec{q}, s)\pi_{a_{\bar{i}} q_{\bar{i}}}^{(m)} \cdot \left[ \hat{R}_{s\vec{a}} + \frac{\gamma}{1-\gamma} \lambda_{q_{\bar{i}}' q_{\bar{i}} o_{\bar{i}}}^{(m)} \lambda_{q_i' q_i o_i}^{(m)} f_{s'sq'\vec{o}\vec{a}m} \right] \right\} \tag{4}$$

$$\lambda_{q'_i q_i o_i}^{*(m)} = \frac{\lambda_{q'_i q_i o_i}^{(m)}}{C_{q_i o_i}} \sum_{s,s',q_{\bar{i}},q'_{\bar{i}},o_{\bar{i}},\vec{a}} \hat{\alpha}^{(m)}(\vec{q},s) \pi_{a_{\bar{i}} q_{\bar{i}}}^{(m)} \pi_{a_i q_i}^{(m)} \lambda_{q'_{\bar{i}} q_{\bar{i}} o_{\bar{i}}}^{(m)} f_{s' s \vec{q'} \vec{o} \vec{a} m} \; . \tag{5}$$

**Note about initialization.** Our initialization procedure (Sections 3.1.1 and 3.1.2) yields deterministic periodic controllers as initializations; a deterministic finite state controller is a stable point of the EM algorithm, since for such a controller the M-step of the EM approach does not change the probabilities. To allow EM to escape the stable point and find even better optima, we add noise to the controllers in order to produce stochastic controllers that can be improved by EM.

## 4  Experiments

Experiments were run for standard POMDP and DEC-POMDP benchmark problems [8, 15, 16, 10] with a time limit of two hours. For both types of benchmarks we ran the proposed infinite-horizon method for deterministic controllers (denoted "Peri") with nine improvement rounds as described in Section 3.1.2. For DEC-POMDP benchmarks we also ran the proposed periodic expectation maximization approach in Section 3.2 (denoted "PeriEM"), initialized by the finite-horizon approach in Section 3.1.1 with nine improvement rounds and the infinite-horizon transformation in Section 3.1.2, paragraph 1. For "PeriEM" a period of 10 was used. For "Peri" a period of 30 was used for problems with discount factor 0.9, 60 for discount factor 0.95, and 100 for larger discount factors. The main comparison methods EM [12] and Mealy NLP [16] (with removal of dominated actions and unreachable state-observation pairs) were implemented using Matlab and the NEOS server was utilized for solving the Mealy NLP non-linear programs. We used the best of parallel experiment runs to choose the number of FSC nodes. EM was run for all problems and Mealy NLP for the Hallway2, decentralized tiger, recycling robots, and wireless network problems. SARSOP [3] was run for all POMDP problems and we also report results from literature [8, 15, 16].

Table 1 shows DEC-POMDP results for the decentralized tiger, recycling robots, meeting in a grid, wireless network [10], co-operative box pushing, and stochastic mars rover problems. A discount factor of 0.99 was used in the wireless network problem and 0.9 in the other DEC-POMDP benchmarks. Table 2 shows POMDP results for the benchmark problems Hallway2, Tag-avoid, Tag-avoid repeat, and Aloha. A discount factor of 0.999 was used in the Aloha problem and 0.95 in the other POMDP benchmarks. Methods whose 95% confidence intervals overlap with that of the best method are shown in bold. The proposed method "Peri" performed best in the DEC-POMDP problems and better than other restricted policy size methods in the POMDP problems. "PeriEM" also performed well, outperforming EM.

## 5  Conclusions and discussion

We introduced a new class of finite state controllers, periodic finite state controllers (periodic FSCs), and presented methods for initialization and policy improvement. In comparisons the resulting methods outperformed state-of-the-art DEC-POMDP and state-of-the-art restricted size POMDP methods and worked very well on POMDPs in general.

In our method the period length was based simply on the discount factor, which already performed very well; even better results could be achieved, for example, by running solutions of different periods in parallel. In addition to the expectation-maximization presented here, other optimization algorithms for infinite-horizon problems could also be adapted to periodic FSCs: for example, the non-linear programming approach [8] could be adapted to periodic FSCs. In brief, a separate value function and separate FSC parameters would be used for each time slice in the periodic FSCs, and the number of constraints would grow linearly with the number of time slices.

#### Acknowledgments

We thank Ari Hottinen for discussions on decision making in wireless networks. The authors belong to the Adaptive Informatics Research Centre (CoE of the Academy of Finland). The work was supported by Nokia, TEKES, Academy of Finland decision number 252845, and in part by the PASCAL2 EU NoE, ICT 216886. This publication reflects the authors' views only.

Table 1: DEC-POMDP benchmarks. Most comparison results are from [8, 15, 16]; we ran EM and Mealy NLP on many of the tests (see Section 4). Note that "Goal-directed" is a special method that can only be applied to problems with goals.

| Algorithm (Size, Time): Value | |
|---|---|
| **DecTiger** | |
| $(\|S\| = 2, \|A_i\| = 3, \|O_i\| = 2)$ | |
| **Peri** $(10 \times 30, 202s)$: | **13.45** |
| PeriEM $(7 \times 10, 6540s)$: | 9.42 |
| Goal-directed $(11, 75s)$: | 5.041 |
| NLP $(19, 6173s)$: | $-1.088$ |
| Mealy NLP $(4, 29s)$: | $-1.49$ |
| EM $(6, 142s)$: | $-16.30$ |
| **Recycling robots** | |
| $(\|S\| = 4, \|A_i\| = 3, \|O_i\| = 2)$ | |
| **Mealy NLP** $(1, 0s)$: | **31.93** |
| Peri $(6 \times 30, 77s)$: | 31.84 |
| PeriEM $(6 \times 10, 272s)$: | 31.80 |
| EM $(2, 13s)$: | 31.50 |
| **Meeting in a 2x2 grid** | |
| $(\|S\| = 16, \|A_i\| = 5, \|O_i\| = 2)$ | |
| **Peri** $(5 \times 30, 58s)$: | **6.89** |
| **PeriEM** $(5 \times 10, 6019s)$: | **6.82** |
| **EM** $(8, 5086s)$: | **6.80** |
| Mealy NLP $(5, 116s)$: | 6.13 |
| HPI+NLP $(7, 16763s)$: | 6.04 |
| NLP $(5, 117s)$: | 5.66 |
| Goal-directed $(4, 4s)$: | 5.64 |
| **Wireless network** | |
| $(\|S\| = 64, \|A_i\| = 2, \|O_i\| = 6)$ | |
| **EM** $(3, 6886s)$: | $-$**175.40** |
| **Peri** $(15 \times 100, 6492s)$: | $-$**181.24** |
| PeriEM $(2 \times 10, 3557s)$: | $-218.90$ |
| Mealy NLP $(1, 9s)$: | $-296.50$ |
| **Box pushing** | |
| $(\|S\| = 100, \|A_i\| = 4, \|O_i\| = 5)$ | |
| **Goal-directed** $(5, 199s)$: | **149.85** |
| **Peri** $(15 \times 30, 5675s)$: | **148.65** |
| Mealy NLP $(4, 774s)$: | 143.14 |
| PeriEM $(4 \times 10, 7164s)$: | 106.68 |
| HPI+NLP $(10, 6545s)$: | 95.63 |
| EM $(6, 7201s)$: | 43.33 |
| **Mars rovers** | |
| $(\|S\| = 256, \|A_i\| = 6, \|O_i\| = 8)$ | |
| **Peri** $(10 \times 30, 6088s)$: | **24.13** |
| Goal-directed $(6, 956s)$: | 21.48 |
| Mealy NLP $(3, 396s)$: | 19.67 |
| PeriEM $(3 \times 10, 7132s)$: | 18.13 |
| EM $(3, 5096s)$: | 17.75 |
| HPI+NLP $(4, 111s)$: | 9.29 |

Table 2: POMDP benchmarks. Most comparison method results are from [16]; we ran EM, SARSOP, and Mealy NLP on one test (see Section 4).

| Algorithm (Size,Time): Value | |
|---|---|
| **Hallway2** | |
| $(\|S\| = 93, \|A\| = 5, \|O\| = 17)$ | |
| **Perseus** $(56, 10s)$: | **0.35** |
| **HSVI2** $(114, 1.5s)$: | **0.35** |
| **PBPI** $(320, 3.1s)$: | **0.35** |
| **SARSOP** $(776, 7211s)$: | **0.35** |
| **HSVI** $(1571, 10010s)$: | **0.35** |
| PBVI $(95, 360s)$: | 0.34 |
| Peri $(160 \times 60, 5252s)$: | 0.34 |
| biased BPI $(60, 790s)$: | 0.32 |
| NLP fixed $(18, 240s)$: | 0.29 |
| NLP $(13, 420s)$: | 0.28 |
| EM $(30, 7129s)$: | 0.28 |
| Mealy NLP $(1, 2s)$: | 0.028 |
| **Tag-avoid** | |
| $(\|S\| = 870, \|A\| = 5, \|O\| = 30)$ | |
| **PBPI** $(818, 1133s)$: | $-$**5.87** |
| SARSOP $(13588, 7394s)$: | $-6.04$ |
| Peri $(160 \times 60, 6394s)$: | $-6.15$ |
| RTDP-BEL $(2.5m, 493s)$: | $-6.16$ |
| Perseus $(280, 1670s)$: | $-6.17$ |
| HSVI2 $(415, 24s)$: | $-6.36$ |
| Mealy NLP $(2, 323s)$: | $-6.65$ |
| biased BPI $(17, 250s)$: | $-6.65$ |
| BPI $(940, 59772s)$: | $-9.18$ |
| NLP $(2, 5596s)$: | $-13.94$ |
| EM $(2, 30s)$: | $-20.00$ |
| **Tag-avoid repeat** | |
| $(\|S\| = 870, \|A\| = 5, \|O\| = 30)$ | |
| **SARSOP** $(15202, 7203s)$: | $-$**10.71** |
| Peri $(160 \times 60, 6316s)$: | $-11.02$ |
| Mealy NLP $(2, 319s)$: | $-11.44$ |
| Perseus $(163, 5656s)$: | $-12.35$ |
| HSVI2 $(8433, 5413s)$: | $-14.33$ |
| NLP $(1, 37s)$: | $-20.00$ |
| EM $(2, 72s)$: | $-20.00$ |
| **Aloha** | |
| $(\|S\| = 90, \|A\| = 29, \|O\| = 3)$ | |
| **SARSOP** $(82, 7201s)$: | **1237.01** |
| **Peri** $(160 \times 100, 6793s)$: | **1236.70** |
| Mealy NLP $(7, 312s)$: | 1221.72 |
| HSVI2 $(5434, 5430s)$: | 1217.95 |
| NLP $(6, 1134s)$: | 1211.67 |
| EM $(3, 7200s)$: | 1120.05 |
| Perseus $(68, 5401s)$: | 853.42 |

## References

[1] R. D. Smallwood and E. J. Sondik. The optimal control of partially observable Markov processes over a finite horizon. *Operations Research*, pages 1071–1088, 1973.

[2] S. Seuken and S. Zilberstein. Formal models and algorithms for decentralized decision making under uncertainty. *Autonomous Agents and Multi-Agent Systems*, 17(2):190–250, 2008.

[3] H. Kurniawati, D. Hsu, and W.S. Lee. Sarsop: Efficient point-based pomdp planning by approximating optimally reachable belief spaces. In *Proc. Robotics: Science and Systems*, 2008.

[4] S. Seuken and S. Zilberstein. Memory-bounded dynamic programming for DEC-POMDPs. In *Proc. of 20th IJCAI*, pages 2009–2016. Morgan Kaufmann, 2007.

[5] F. Wu, S. Zilberstein, and X. Chen. Point-based policy generation for decentralized POMDPs. In *Proc. of 9th AAMAS*, pages 1307–1314. IFAAMAS, 2010.

[6] P. Poupart and C. Boutilier. Bounded finite state controllers. *Advances in neural information processing systems*, 16:823–830, 2003.

[7] M. Toussaint, S. Harmeling, and A. Storkey. Probabilistic inference for solving (PO)MDPs. Technical report, University of Edinburgh, 2006.

[8] C. Amato, D. Bernstein, and S. Zilberstein. Optimizing Memory-Bounded Controllers for Decentralized POMDPs. In *Proc. of 23rd UAI*, pages 1–8. AUAI Press, 2007.

[9] A. Kumar and S. Zilberstein. Point-Based Backup for Decentralized POMDPs: Complexity and New Algorithms. In *Proc. of 9th AAMAS*, pages 1315–1322. IFAAMAS, 2010.

[10] Joni Pajarinen and Jaakko Peltonen. Efficient Planning for Factored Infinite-Horizon DEC-POMDPs. In *Proc. of 22nd IJCAI*, pages 325–331. AAAI Press, July 2011.

[11] D. S. Bernstein, R. Givan, N. Immerman, and S. Zilberstein. The Complexity of Decentralized Control of Markov Decision Processes. *Mathematics of Operations Research*, 27(4):819–840, 2002.

[12] A. Kumar and S. Zilberstein. Anytime Planning for Decentralized POMDPs using Expectation Maximization. In *Proc. of 26th UAI*, 2010.

[13] D.S. Bernstein, E.A. Hansen, and S. Zilberstein. Bounded policy iteration for decentralized POMDPs. In *Proc. of 19th IJCAI*, pages 1287–1292. Morgan Kaufmann, 2005.

[14] D. Szer and F. Charpillet. An optimal best-first search algorithm for solving infinite horizon DEC-POMDPs. *Proc. of 16th ECML*, pages 389–399, 2005.

[15] C. Amato and S. Zilberstein. Achieving goals in decentralized POMDPs. In *Proc. of 8th AAMAS*, volume 1, pages 593–600. IFAAMAS, 2009.

[16] C. Amato, B. Bonet, and S. Zilberstein. Finite-State Controllers Based on Mealy Machines for Centralized and Decentralized POMDPs. In *Proc. of 24th AAAI*, 2010.

[17] S. Ji, R. Parr, H. Li, X. Liao, and L. Carin. Point-based policy iteration. In *Proc. of 22nd AAAI*, volume 22, page 1243, 2007.

[18] F.A. Oliehoek, M.T.J. Spaan, and N. Vlassis. Optimal and approximate q-value functions for decentralized pomdps. *Journal of Artificial Intelligence Research*, 32(1):289–353, 2008.

